# Efficient and direct estimation of a neural subunit model for sensory coding

**Brett Vintch**    **Andrew D. Zaharia**    **J. Anthony Movshon**    **Eero P. Simoncelli** [†]

Center for Neural Science, and
[†]Howard Hughes Medical Institute
New York University
New York, NY 10003
vintch@cns.nyu.edu

## Abstract

Many visual and auditory neurons have response properties that are well explained by pooling the rectified responses of a set of spatially shifted linear filters. These filters cannot be estimated using spike-triggered averaging (STA). Subspace methods such as spike-triggered covariance (STC) can recover multiple filters, but require substantial amounts of data, and recover an orthogonal basis for the subspace in which the filters reside rather than the filters themselves. Here, we assume a linear-nonlinear–linear-nonlinear (LN-LN) cascade model in which the first linear stage is a set of shifted ('convolutional') copies of a common filter, and the first nonlinear stage consists of rectifying scalar nonlinearities that are identical for all filter outputs. We refer to these initial LN elements as the 'subunits' of the receptive field. The second linear stage then computes a weighted sum of the responses of the rectified subunits. We present a method for directly fitting this model to spike data, and apply it to both simulated and real neuronal data from primate V1. The subunit model significantly outperforms STA and STC in terms of cross-validated accuracy and efficiency.

## 1   Introduction

Advances in sensory neuroscience rely on the development of testable functional models for the encoding of sensory stimuli in neural responses. Such models require procedures for fitting their parameters to data, and should be interpretable in terms both of sensory function and of the biological elements from which they are made. The most common models in the visual and auditory literature are based on linear-nonlinear (LN) cascades, in which a linear stage serves to project the high-dimensional stimulus down to a one-dimensional signal, where it is then nonlinearly transformed to drive spiking. LN models are readily fit to data, and their linear operators specify the stimulus selectivity and invariance of the cell. The weights of the linear stage may be loosely interpreted as representing the efficacy of synapses, and the nonlinearity as a transformation from membrane potential to firing rate.

For many visual and auditory neurons, responses are not well described by projection onto a single linear filter, but instead reflect a combination of several filters. In the cat retina, the responses of Y cells have been described by linear pooling of shifted rectified linear filters, dubbed "subunits" [1, 2]. Similar behaviors are seen in guinea pig [3] and monkey retina [4]. In the auditory nerve, responses are described as computing the envelope of the temporally filtered sound waveform, which can be computed via summation of squared quadrature filter responses [5]. In primary visual cortex (V1), simple cells are well described using LN models [6, 7], but complex cell responses are more like a

superposition of multiple spatially shifted simple cells [8], each with the same orientation and spatial frequency preference [9]. Although the description of complex cells is often reduced to a sum of two squared filters in quadrature [10], more recent experiments indicate that these cells (and indeed most 'simple' cells) require multiple shifted filters to fully capture their responses [11, 12, 13]. Intermediate nonlinearities are also required to describing the response properties of some neurons in V2 to stimuli (e.g., angles [14] and depth edges [15]).

Each of these examples is consistent with a canonical but constrained LN-LN model, in which the first linear stage consists of convolution with one (or a few) filters, and the first nonlinear stage is point-wise and rectifying. The second linear stage then pools the responses of these "subunits" using a weighted sum, and the final nonlinearity converts this to a firing rate. Hierarchical stacks of this type of "generalized complex cell" model have also been proposed for machine vision [16, 17]. What is lacking is a method for validating this model by fitting it directly to spike data.

A widely used procedure for fitting a simple LN model to neural data is reverse correlation [18, 19]. The spike-triggered average of a set of Gaussian white noise stimuli provides an unbiased estimate of the linear kernel. In a subunit model, the initial linear stage projects the stimulus into a multi-dimensional subspace, which can be estimated using spike-triggered covariance (STC) [20, 21]. This has been used successfully for fly motion neurons [22], vertebrate retina [23], and primary visual cortex [24, 11]. But this method relies on a Gaussian stimulus ensemble, requires a substantial amount of data, and recovers only a set of orthogonal axes for the response subspace—not the underlying biological filters. More general methods based on information maximization alleviate some of the stimulus restrictions [25] but strongly limit the dimensionality of the recoverable subspace and still produce only a basis for the subspace.

Here, we develop a specific subunit model and a maximum likelihood procedure to estimate its parameters from spiking data. We fit the model to both simulated and real V1 neuronal data, demonstrating that it is substantially more accurate for a given amount of data than the current state-of-the-art V1 model which is based on STC [11], and that it produces biologically interpretable filters.

## 2 Subunit model

We assume that neural responses arise from a weighted sum of the responses of a set of nonlinear subunits. Each subunit applies a linear filter to its input (which can be either the raw stimulus, or the responses arising from a previous stage in a hierarchical cascade), and transforms the filtered response using a memoryless rectifying nonlinearity. A critical simplification is that the subunit filters are related by a fixed transformation; here, we assume they are spatially translated copies of a common filter, and thus the population of subunits can be viewed as computing a convolution. For example, the subunits of a V1 complex cell could be simple cells in V1 that share the same orientation and spatial frequency preference, but differ in spatial location, as originally proposed by Hubel & Wiesel [8, 9]. We also assume that all subunits use the same rectifying nonlinearity. The response to input defined over two discrete spatial dimensions and time, $x(i, j, t)$, is written as:

$$\hat{r}(t) = \sum_{m,n} w_{m,n} \, f_\Theta \left( \sum_{i,j,\tau} k(m, n, \tau) \cdot x(i - m, j - n, t - \tau) \right) + \ldots + b, \qquad (1)$$

where $k$ is the subunit filter, $f_\Theta$ is a point-wise function parameterized by vector $\Theta$, $w_{n,m}$ are the spatial weights, and $b$ is an additive baseline. The ellipsis indicates that we allow for multiple subunit channels, each with its own filter, nonlinearity, and pooling weights. We interpret $\hat{r}(t)$ as a 'generator potential', (e.g., time-varying membrane voltage) which is converted to a firing rate by another rectifying nonlinearity.

The subunit model of Eq. (1) may be seen as a specific instance of a *subspace model*, in which the input is initially projected onto a linear subspace. Bialek and colleagues introduced spike-triggered covariance as a means of recovering such subspaces [20, 22]. Specifically, eigenvector analysis of the covariance matrix of the spike-triggered input ensemble exposes orthogonal axes for which the spike-triggered ensemble has a variance that differs significantly from that of the raw input ensemble. These axes may be separated into those along which variance is greater (excitatory) or less than (suppressive) that of the input. Figure 1 demonstrates what happens when STC is applied to a simulated complex cell with 15 spatially shifted subunits. The response of this model cell is

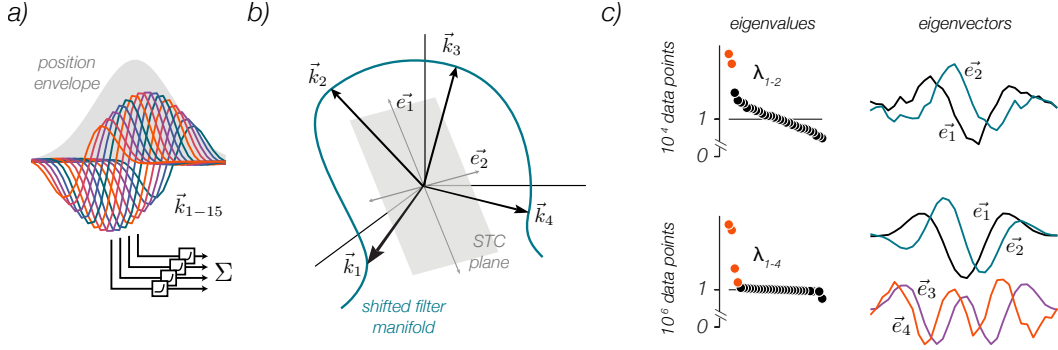

Figure 1: Spike-triggered covariance analysis of a simulated V1 complex cell. (a) The model output is formed by summing the rectified responses of multiple linear filter kernels which are shifted and scaled copies of a canonical form. (b) The shifted filters lie along a manifold in stimulus space (four shown), and are not mutually orthogonal in general. STC recovers an orthogonal basis for a low-dimensional subspace that contains this manifold by finding the directions in stimulus space along which spikes are elicited or suppressed. (c) STC analysis of this model cell returns a variable number of filters dependent upon the amount of acquired data. A modest amount of data typically reveals two strong STC eigenvalues (top), whose eigenvectors form a quadrature (90-degree phase-shifted) pair and span the best-fitting plane for the set of shifted model filters. These will generally have tuning properties (orientation, spatial frequency) similar to the true model filters. However, the manifold does not generally lie in a two-dimensional subspace [26], and a larger data set reveals additional eigenvectors (bottom) that serve to capture the deviations from the $\vec{e}_{1,2}$ plane. Due to the constraint of mutual orthogonality, these filters are usually not localized and they have tuning properties that differ from true model filters.

$\hat{r}(t) = \sum_i w_i \lfloor (\vec{k}_i \cdot \vec{x}(t)) \rfloor^2$, where the $\vec{k}$'s are shifted filters, $w$ weights filters by position, and $\vec{x}$ is the stimulus vector. The recovered STC axes span the same subspace as the shifted model filters, but there are fewer of them, and the enforced orthogonality of eigenvectors means that they are generally not a direct match to any of the model filters. This has also been observed in filters extracted from physiological data [11, 12]. Although one may follow the STC analysis by indirectly identifying a localized filter whose shifted copies span the recovered subspace [11, 13], the reliance on STC still imposes the stimulus limitations and data requirements mentioned above.

## 3   Direct subunit model estimation

A generic subspace method like STC does not exploit the specific structure of the subunit model. We therefore developed an estimation procedure explicitly tailored for this type of computation. We first introduce a piecewise-linear parameterization of the subunit nonlinearity:

$$f(s) = \sum_l \alpha_l T_l(s), \tag{2}$$

where the $\alpha$'s scale a small set of overlapping 'tent' functions, $T_l(\cdot)$, that represent localized portions of $f(\cdot)$ (we find that a dozen or so basis functions are typically sufficient to provide the needed flexibility). Incorporating this into the model response of Eq. (1) allows us to fold the second linear pooling stage and the subunit nonlinearity into a single sum:

$$\hat{r}(t) = \sum_{m,n,l} w_{m,n} \alpha_l\, T_l \left( \sum_{i,j,\tau} k(m,n,\tau) \cdot x(i-m, j-n, t-\tau) \right) + ... + b. \tag{3}$$

The model is now partitioned into two linear stages, separated by the *fixed* nonlinear functions $T_l(\cdot)$. In the first, the stimulus is convolved with $k$, and in the second, the nonlinear responses are summed with a set of weights that are separable in the indices $l$ and $n, m$. The partition motivates the use of an iterative coordinate descent scheme: the linear weights of each portion are optimized in alternation,

while the other portion is held constant. For each step, we minimized the mean square error between the observed firing rate of a cell and the firing rate predicted by the model. For models that include two subunit channels we optimize over both channels simultaneously (see section 3.3 for comments regarding two-channel initialization).

## 3.1 Estimating the convolutional subunit kernel

The first coordinate descent leg optimizes the convolutional subunit kernel, $k$, using gradient descent while fixing the subunit nonlinearity and the final linear pooling. Because the tent basis functions are fixed and piecewise linear, the gradient is easily determined. This property also ensures that the descent is locally convex: assuming that updating $k$ does not cause any of the the linear subunit responses to jump between the localized tent functions representing $f$, then the optimization is linear and the objective function is quadratic. In practice, the full gradient descent path causes the linear subunit responses to move slowly across bins of the piecewise nonlinearity. However, we include a regularization term to impose smoothness on the nonlinearity (see below) and this yields a well-behaved minimization problem for $k$.

## 3.2 Estimating the subunit nonlinearities and linear subunit pooling

The second leg of coordinate descent optimizes the subunit nonlinearity (more specifically, the weights on the tent functions, $\alpha_l$), and the subunit pooling, $w_{n,m}$. As described above, the objective is bilinear in $\alpha_l$ and $w_{n,m}$ when $k$ is fixed. Estimating both $\alpha_l$ and $w_{n,m}$ can be accomplished with alternating least-squares, which assures convergence to a (local) minimum [27]. We also include two regularization terms in the objective function. The first ensures smoothness in the nonlinearity $f$, by penalizing the square of the second derivative of the function in the least-squares fit. This smooth nonlinearity helps to guarantee that the optimization of $k$ is well behaved, even where finite data sets leave the function poorly constrained. We also include a cross-validated ridge prior for the pooling weights to bias $w_{n,m}$ toward zero. The filter kernel $k$ can also be regularized to ensure smoothness, but for the examples shown here we did not find the need to include such a term.

## 3.3 Model initialization

Our objective function is non-convex and contains local minima, so the selection of initial parameter values may affect the solution. We found that initializing our two-channel subunit model to have a positive pooling function for one channel and a negative pooling function for the second channel allowed the optimization of the second channel to proceed much more quickly. This is probably due in part to a suppressive channel that is much weaker than the excitatory channel in general. We initialized the nonlinearity to halfwave-rectification for the excitatory channel and fullwave-rectification for the suppressive channel.

To initialize the convolutional filter we use a novel technique that we term 'convolutional STC'. The subunit model describes a receptive field as the linear combination of nonlinear kernel responses that spatially tile the stimulus. Thus, the contribution of each localized patch of stimulus (of a size equal to the subunit kernel) is the same, up to a scale factor set by the weighting used in the subsequent pooling stage. As such, we compute an STC analysis on the union of all localized patches of stimuli. For each subunit location, $\{m, n\}$, we extract the local stimulus values in a window, $g_{m,n}(i, j)$, the size of the convolutional kernel and append them vertically in a 'local' stimulus matrix. As an initial guess for the pooling weights, we weight each of these blocks by a Gaussian spatial profile, chosen to roughly match the size of the receptive field. We also generate a vector containing the vertical concatenation of copies of the measured spike train, $\vec{r}$ (one copy for each subunit location).

$$\begin{pmatrix} w_{1,1} X_{g_{1,1}(i,j)} \\ w_{1,2} X_{g_{1,2}(i,j)} \\ \vdots \end{pmatrix} \rightarrow X_{loc} \; ; \quad \begin{pmatrix} \vec{r} \\ \vec{r} \\ \vdots \end{pmatrix} \rightarrow \vec{r}_{loc}. \tag{4}$$

After performing STC analysis on the localized stimulus matrix, we use the first (largest variance) eigenvector to initialize the subunit kernel of the excitatory channel, and the last (lowest variance) eigenvector to initialize the kernel of the suppressive channel. In practice, we find that this initialization greatly reduces the number of iterations, and thus the run time, of the optimization procedure.

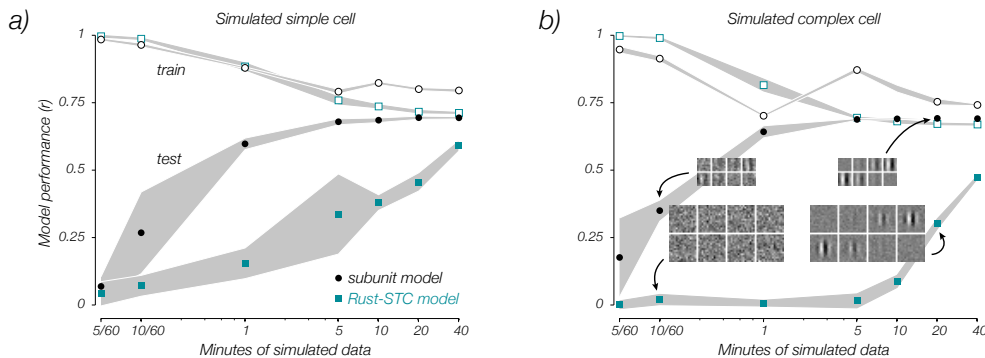

Figure 2: Model fitting performance for simulated V1 neurons. Shown are correlation coefficients for the subunit model (black circles) and the Rust-STC model (blue squares) [11], computed on both the training data (open), and on a holdout test set (closed). Spike counts for each presented stimulus frame are drawn from a Poisson distribution. Shaded regions indicate $\pm$ 1 s.d. for 5 simulation runs. (a) 'Simple' cell, with spike rate determined by the halfwave-rectified and squared response of a single oriented linear filter. (b) 'Complex' cell, with rate determined by a sum of squared Gabor filters arranged in spatial quadrature. Insets show estimated filters for the subunit (top) and Rust-STC (bottom) models with ten seconds (400 frames; left) and 20 minutes (48,000 frames; right) of data.

## 4 Experiments

We fit the subunit model to physiological data sets in 3 different primate cortical areas: V1, V2, and MT. The model is able to explain a significant amount of variance for each of these areas, but for illustrative purposes we show here only data for V1. Initially, we use simulated V1 cells to compare the performance of the subunit model to that of the Rust-STC model [11], which is based upon STC analysis.

### 4.1 Simulated V1 data

We simulated the responses of canonical V1 simple cells and complex cells in response to white noise stimuli. Stimuli consisted of a 16x16 spatial array of pixels whose luminance values were set to independent ternary white noise sequences, updated every 25 ms (or 40 Hz). The simulated cells use spatiotemporally oriented Gabor filters: The simple cell has one even-phase filter and a half-squaring output nonlinearity while the complex cell has two filters (one even and one odd) whose squared responses are combined to give a firing rate. Spike counts are drawn from a Poisson distribution, and overall rates are scaled so as to yield an average of 40 ips (i.e. one spike per time bin).

For consistency with the analysis of the physiological data, we fit the simulated data using a subunit model with two subunit channels (even though the simulated cells only possess an excitatory channel). When fitting the Rust-STC model, we followed the procedure described in [11]. Briefly, after the STA and STC filters are estimated, they are weighted according to their predictive power and combined in excitatory and suppressive pools, $E$ and $S$ (we use cross-validation to determine the number of filters to use for each pool). These two pooled responses are then combined using a joint output nonlinearity: $\hat{r}(t)_{Rust} = \alpha + (\beta E^\rho - \delta S^\rho)/(\gamma E^\rho + \epsilon S^\rho + 1)$. Parameters $\{\alpha, \beta, \delta, \gamma, \epsilon, \rho\}$ are optimized to minimizing mean squared error between observed spike counts and the model rate.

Model performances, measured as the correlation between the model rate and spike count, are shown in Figure 2. In low data regimes, both models perform nearly perfectly on the training data, but poorly on separate test data not used for fitting, a clear indication of over-fitting. But as the data set increases in size, the subunit model rapidly improves, reaching near-perfect performance for modest spike counts. The Rust-STC model also improves, but much more slowly; It requires more than an order of magnitude more data to achieve the same performance as the subunit model. This

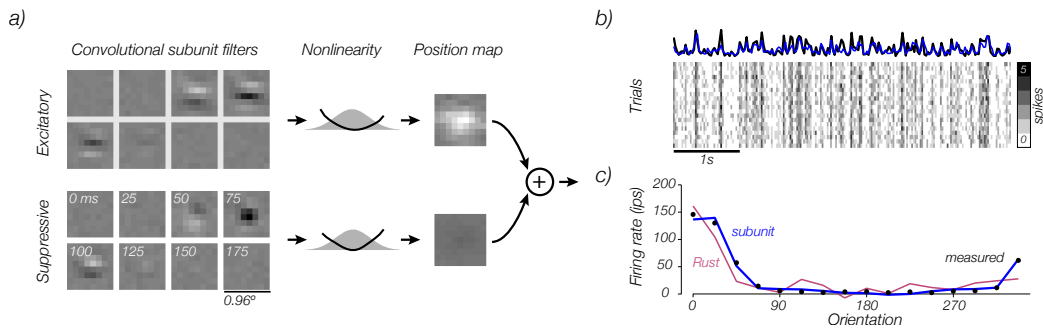

Figure 3: Two-channel subunit model fit to a physiological data from a macaque V1 cell. (a) Fitted parameters for the excitatory (top row) and suppressive (bottom row) channels, including the space-time subunit filters (8 grayscale images, corresponding to different time frames), the nonlinearity, and the spatial weighting function $w_{n,m}$ that is used to combine the subunit responses. (b) A raster showing spiking responses to 20 repeated presentations of an identical stimulus with the average spike count (black) and model prediction (blue) plotted above. (c) Simulated models (subunit model: blue, Rust-STC model: purple) and measured (black) responses to drifting sinusoidal gratings.

inefficiency is more pronounced for the complex cell, because the simple cell is fully explained by the STA filter, which can be estimated much more reliably than the STC filters for small amounts of data. We conclude that directly fitting the subunit model is much more efficient in the use of data than using STC to estimate a subspace model.

## 4.2 Physiological data from macaque V1

We presented spatio-temporal pixel noise to 38 cells recorded from V1 in anesthetized macaques (see [11] for details of experimental design). The stimulus was a 16x16 grid with luminance values set by independent ternary white noise sequences refreshed at 40 Hz. For 21 neurons we also presented 20 repeats of a sequence of 1000 stimulus frames as a validation set. The model filters were assumed to respond over a 200 ms (8 frame) causal time window in which the stimulus most strongly affected the firing of the neurons, and thus, model responses were derived from a stimulus vector with 2048 dimensions (16x16x8).

Figure 3 shows the fit of a 2-channel subunit model to data from a typical V1 cell. Figure 3a illustrates the subunit kernels and their associated nonlinearities and spatial pooling maps, for both the excitatory channel (top row) and the suppressive channel (bottom row). The two channels show clear but opposing direction selectivity, starting at a latency of 50 ms. The fact that this cell is complex is reflected in two aspects of the model parameters. First, the model shows a symmetric, full-wave rectifying nonlinearity for the excitatory channel. Second, the final linear pooling for this channel is diffuse over space, eliciting a response that is invariant to the exact spatial position and phase of the stimulus.

For this particular example the model fits well. For the cross-validated set of repeated stimuli (which have the same structure as for the fitting data), on average the model correlates with each trial's firing rate with an r-value of 0.54. A raster of spiking responses to twenty repetitions of a 5 s stimulus are depicted in Fig. 3b, along with the average firing rate and the model prediction, which are well matched. The model can also capture the direction selectivity of this cell's response to moving sinusoidal gratings (whose spatial and temporal frequency are chosen to best drive the cell) (Fig. 3c). The subunit model acceptably fits most of the cells we recorded in V1. Moreover, fit quality is not correlated with modulation index ($r = -0.08$; $n.s.$), suggesting that the model captures the behavior of both simple and complex cells equally well.

The fitted subunit model also significantly outperforms the Rust-STC model in terms of predicting responses to novel data. Figure 4a shows the performance of the Rust-STC and subunit models for 21 V1 neurons, for both training data and test data on *single trials*. For the training data, the Rust-

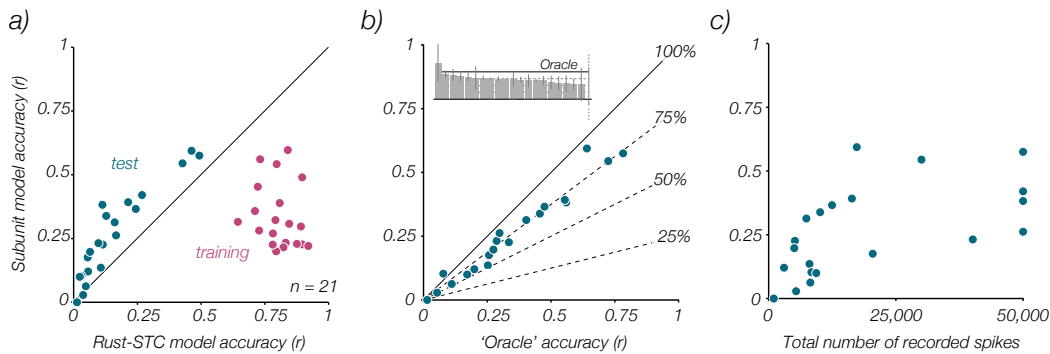

Figure 4: Model performance comparisons on physiological data. (a) Subunit model performance vs. Rust-STC model for V1 data. Training accuracy is computed for a single variable-length sequence extracted from the fitting data. Test accuracy is computed on the average response to 20 repeats of a 25 s stimulus. (b) Subunit model performance vs. an 'Oracle' model for V1 data (see text). Each point represents the average accuracy in predicting responses to each of 20 repeated stimuli. The oracle model uses the average spike count over the other 19 repeats as a prediction. Inset: Ratio of subunit-to-oracle performance. Error bars indicate 1 s.d. (c) Subunit model performance on test data, as a function of the total number of recorded spikes.

STC model performs significantly better than the subunit model (Figure 4a; $< r_{Rust} >= 0.81$, $< r_{subunit} >= 0.33$; $p \ll 0.005$). However, this is primarily due to over-fitting: Visual inspection of the STC kernels for most cells reveals very little structure. For test data (that was not included in the data used to fit the models), the subunit model exhibits significantly better performance than the Rust-STC model ($< r_{Rust} >= 0.16$, $< r_{subunit} >= 0.27$; $p \ll 0.005$). This is primarily due to over-fitting in the STC analysis. For a stimulus composed of a 16x16 pixel grid with 8 frames, the spike-triggered covariance matrix contains over 2 million parameters. For the same stimulus, a subunit model with two channels and an 8x8x8 subunit kernel has only about 1200 parameters.

The subunit model performs well when compared to the Rust-STC model, but we were interested in obtaining a more absolute measure of performance. Specifically, no purely stimulus-driven model can be expected to explain the response variability seen across repeated presentations of the same stimulus. We can estimate an upper bound on stimulus-driven model performance by implementing an empirical 'oracle' model that uses the average response over all but one of a set of repeated stimulus trials to predict the response on the remaining trial. Over the 21 neurons with repeated stimulus data, we found that the subunit model achieved, on average, 76% the performance of the oracle model (Figure 4b). Moreover, the cells that were least well fit by the subunit model were also the cells that responded only weakly to the stimulus (Figure 4c). We conclude that, for most cells, the fitted subunit model explains a significant fraction of the response that can be explained by any stimulus-driven model.

## 5    Discussion

Subunits have been proposed as a qualitative description of many types of receptive fields in sensory systems [2, 28, 8, 11, 12], and have enjoyed a recent renewal of interest by the modeling community [13, 29]. Here we have described a new parameterized canonical subunit model that can be applied to an arbitrary set of inputs (either a sensory stimulus, or a population of afferents from a previous stage of processing), and we have developed a method for directly estimating the parameters of this model from measured spiking data. Compared with STA or STC, the model fits are more accurate for a given amount of data, less sensitive to the choice of stimulus ensemble, and more interpretable in terms of biological mechanism.

For V1, we have applied this model directly to the visual stimuli, adopting the simplifying assumption that subcortical pathways faithfully relay the image data to V1. Higher visual areas build their responses on the afferent inputs arriving from lower visual areas, and we have applied this subunit

model to such neurons by first simulating the responses of a population of the afferent V1 neurons, and then optimizing a subunit model that best maps these afferent responses to the spiking responses observed in the data. Specifically, for neurons in area V2, we model the afferent V1 population as a collection of simple cells that tile visual space. The V1 filters are chosen to uniformly cover the space of orientations, scales, and positions [30]. We also include four different phases. For neurons in area MT (V5), we use an afferent V1 population that also includes direction selective subunits, because the projections from V1 to MT are known to be sensitive to the direction of visual motion [31]. Specifically, the V1 filters are a rotation-invariant set of 3-dimensional, space-space-time steerable filters [32]. We fit these models to neural responses to textured stimuli that varied in contrast and local orientation content (for MT, the local elements also drift over time). Our preliminary results show that the subunit model outperforms standard models for these higher order areas as well.

We are currently working to refine and generalize the subunit model in a number of ways. The mean squared error objective function, while computationally appealing, does not accurately reflect the noise properties of real neurons, whose variance changes with their mean rate. A likelihood objective function, based on a Poisson or similar spiking model, can improve the accuracy of the fitted model, but it does so at a cost to the simplicity of model estimation (e.g. Alternating Least Squares can no longer be used to solve the bilinear problem). Real neurons also possess other forms of nonlinearities, such as local gain control that is been observed in neurons through the visual and auditory systems [33]. We are exploring means by which this functionality can be included directly in the model framework (e.g. [11]), while retaining the tractability of the parameter estimation.

### Acknowledgments

This work was supported by the Howard Hughes Medical Institute, and by NIH grant EY04440.

# References

[1] H. B. Barlow and W. R. Levick. The mechanism of directionally selective units in rabbit's retina. *The Journal of Physiology*, 178(3):477, June 1965.

[2] S. Hochstein and R. M. Shapley. Linear and nonlinear spatial subunits in Y cat retinal ganglion cells, 1976.

[3] J. B. Demb, K. Zaghloul, L. Haarsma, and P. Sterling. Bipolar cells contribute to nonlinear spatial summation in the brisk-transient (Y) ganglion cell in mammalian retina. *The Journal of neuroscience*, 21(19):7447–7454, 2001.

[4] J.D. Crook, B.B. Peterson, O.S. Packer, F.R. Robinson, J.B. Troy, and D.M. Dacey. Y-cell receptive field and collicular projection of parasol ganglion cells in macaque monkey retina. *The Journal of neuroscience*, 28(44):11277–11291, 2008.

[5] P.X. Joris, C.E. Schreiner, and A. Rees. Neural processing of amplitude-modulated sounds. *Physiol. Rev.*, 84:541–577, 2004.

[6] J. P. Jones and L. A. Palmer. The two-dimensional spatial structure of simple receptive fields in cat striate cortex. *Journal of neurophysiology*, 58(6):1187–1211, 1987.

[7] G. C. DeAngelis, I. Ohzawa, and R. D. Freeman. Spatiotemporal organization of simple-cell receptive fields in the cat's striate cortex. I. General characteristics and postnatal development. *Journal of neurophysiology*, 69(4):1091–1117, 1993.

[8] D. H. Hubel and T. N. Wiesel. Receptive fields, binocular interaction and functional architecture in the cat's visual cortex. *The Journal of Physiology*, 160(1):106–154, 1962.

[9] J. A. Movshon, I. D. Thompson, and D. J. Tolhurst. Receptive field organization of complex cells in the cat's striate cortex. *The Journal of Physiology*, 283(1):79–99, 1978.

[10] E. H. Adelson and J. R. Bergen. Spatiotemporal energy models for the perception of motion. *Journal of the Optical Society of America A*, 2(2):284–299, 1985.

[11] N. C. Rust, O. Schwartz, J. A. Movshon, and E. P. Simoncelli. Spatiotemporal elements of macaque V1 receptive fields. *Neuron*, 46(6):945–956, June 2005.

[12] X. Chen, F. Han, M. M. Poo, and Y. Dan. Excitatory and suppressive receptive field subunits in awake monkey primary visual cortex (V1). *Proceedings of the National Academy of Sciences*, 104(48):19120–19125, November 2007.

[13] T. Lochmann, T. Blanche, and D. A. Butts. Construction of direction selectivity in V1: from simple to complex cells. *Computational and Systems Neuroscience (CoSyNe)*, 2011.

[14] M. Ito and H. Komatsu. Representation of angles embedded within contour stimuli in area V2 of macaque monkeys. *The Journal of neuroscience*, 24(13):3313–3324, 2004.

[15] C. E. Bredfeldt, J. C. A. Read, and B. G. Cumming. A quantitative explanation of responses to disparity-defined edges in macaque V2. *Journal of neurophysiology*, 101(2):701–713, 2009.

[16] K. Fukushima. Neocognitron: A self-organizing neural network model for a mechanism of pattern recognition unaffected by shift in position. *Biological cybernetics*, 36(4):193–202, 1980.

[17] M. Riesenhuber and T. Poggio. Hierarchical models of object recognition in cortex. *Nature neuroscience*, 2:1019–1025, 1999.

[18] E. De Boer. *Reverse correlation I. A heuristic introduction to the technique of triggered correlation with application to the analysis of compound systems*. Proc. Kon. Nederl. Akad. Wet, 1968.

[19] E. J. Chichilnisky. A simple white noise analysis of neuronal light responses. *Network: Computation in Neural Systems*, 12(2):199–213, 2001.

[20] R. D. R. V. Steveninck and W. Bialek. Real-Time Performance of a Movement-Sensitive Neuron in the Blowfly Visual System: Coding and Information Transfer in Short Spike Sequences. *Proceedings of the Royal Society B: Biological Sciences*, 234(1277):379–414, September 1988.

[21] O. Schwartz, J. W. Pillow, N.C. Rust, and E.P. Simoncelli. Spike-triggered neural characterization. *Journal of Vision*, 6(4):13–13, February 2006.

[22] N. Brenner, W. Bialek, and R. de Ruyter van Steveninck. Adaptive rescaling maximizes information transmission. *Neuron*, 26(3):695–702, June 2000.

[23] O. Schwartz, E. J. Chichilnisky, and E. P. Simoncelli. Characterizing neural gain control using spike-triggered covariance. *Advances in neural information processing systems*, 1:269–276, 2002.

[24] J. Touryan, B. Lau, and Y Dan. Isolation of relevant visual features from random stimuli for cortical complex cells. *The Journal of neuroscience*, 22(24):10811–10818, 2002.

[25] T. Sharpee, N. C. Rust, and W. Bialek. Analyzing neural responses to natural signals: maximally informative dimensions. *Neural computation*, 16(2):223–250, 2004.

[26] C. Ekanadham, D. Tranchina, and E. P. Simoncelli. Recovery of sparse translation-invariant signals with continuous basis pursuit. *IEEE Trans Signal Processing*, 59(10):4735–4744, Oct 2011.

[27] M. Ahrens, L. Paninski, and M. Sahani. Inferring input nonlinearities in neural encoding models. *Network: Computation in Neural Systems*, 19(1):35–67, 2008.

[28] J. D. Victor and R. M. Shapley. The nonlinear pathway of Y ganglion cells in the cat retina. *The Journal of General Physiology*, 74(6):671–689, December 1979.

[29] M. Eickenberg, R. J. Rowekamp, M. Kouh, and T. O. Sharpee. Characterizing responses of translation-invariant neurons to natural stimuli: maximally informative invariant dimensions. *Neural computation*, 24(9):2384–2421, September 2012.

[30] E. P. Simoncelli and W. T. Freeman. The steerable pyramid: A flexible architecture for multi-scale derivative computation. *Image Processing, 1995. Proceedings., International Conference on*, 3:444–447 vol. 3, 1995.

[31] J. A. Movshon and W. T. Newsome. Visual response properties of striate cortical neurons projecting to area MT in macaque monkeys. *The Journal of neuroscience*, 16(23):7733–7741, 1996.

[32] E. P. Simoncelli and D. J. Heeger. A model of neuronal responses in visual area MT. *Vision Research*, 38(5):743–761, March 1998.

[33] M. Carandini and D. J. Heeger. Normalization as a canonical neural computation. *Nature Reviews Neuroscience*, 13(1):51–62, November 2011.

